# $\nu$-Arc: Ensemble Learning in the Presence of Outliers

**G. Rätsch[†], B. Schölkopf[‡], A. Smola[*],**
**K.-R. Müller[†], T. Onoda[††], and S. Mika[†]**
† GMD FIRST, Rudower Chaussee 5, 12489 Berlin, Germany
‡ Microsoft Research, 1 Guildhall Street, Cambridge CB2 3NH, UK
* Dep. of Engineering, ANU, Canberra ACT 0200, Australia
†† CRIEPI, 2-11-1, Iwado Kita, Komae-shi, Tokyo, Japan
{raetsch, klaus, mika}@first.gmd.de, bsc@microsoft.com,
Alex.Smola@anu.edu.au, onoda@criepi.denken.or.jp

## Abstract

AdaBoost and other ensemble methods have successfully been applied to a number of classification tasks, seemingly defying problems of overfitting. AdaBoost performs gradient descent in an error function with respect to the margin, asymptotically concentrating on the patterns which are hardest to learn. For very noisy problems, however, this can be disadvantageous. Indeed, theoretical analysis has shown that the margin *distribution*, as opposed to just the minimal margin, plays a crucial role in understanding this phenomenon. Loosely speaking, some outliers should be tolerated if this has the benefit of substantially increasing the margin on the remaining points. We propose a new boosting algorithm which allows for the possibility of a pre-specified fraction of points to lie in the margin area or even on the wrong side of the decision boundary.

## 1 Introduction

Boosting and related Ensemble learning methods have been recently used with great success in applications such as Optical Character Recognition (e.g. [8, 16]).

The idea of a large minimum margin [17] explains the good generalization performance of AdaBoost in the low noise regime. However, AdaBoost performs worse on noisy tasks [10, 11], such as the *iris* and the *breast cancer* benchmark data sets [1]. On the latter tasks, a large margin on *all* training points cannot be achieved without adverse effects on the generalization error. This experimental observation was supported by the study of [13] where the generalization error of ensemble methods was bounded by the sum of the fraction of training points which have a margin smaller than some value $\rho$, say, plus a complexity term depending on the base hypotheses and $\rho$. While this bound can only capture part of what is going on in practice, it nevertheless already conveys the message that in some cases it pays to allow for some points which have a small margin, or are misclassified, if this leads to a larger overall margin on the remaining points.

To cope with this problem, it was mandatory to construct *regularized* variants of AdaBoost, which traded off the number of margin errors and the size of the margin

[9, 11]. This goal, however, had so far been achieved in a heuristic way by introducing regularization parameters which have no immediate interpretation and which cannot be adjusted easily.

The present paper addresses this problem in two ways. Primarily, it makes an *algorithmic* contribution to the problem of constructing regularized boosting algorithms. However, compared to the previous efforts, it parameterizes the above trade-off in a much more intuitive way: its only free parameter directly determines the fraction of margin errors. This, in turn, is also appealing from a *theoretical* point of view, since it involves a parameter which controls a quantity that plays a crucial role in the generalization error bounds (cf. also [9, 13]). Furthermore, it allows the user to roughly specify this parameter once a reasonable estimate of the expected error (possibly from other studies) can be obtained, thus reducing the training time.

## 2    Boosting and the Linear Programming Solution

Before deriving a new algorithm, we briefly discuss the properties of the solution generated by standard AdaBoost and, closely related, Arc-GV [2], and show the relation to a linear programming (LP) solution over the class of base hypotheses $G$. Let $\{g_t(\mathbf{x}) : t = 1, \ldots, T\}$ be a sequence of hypotheses and $\boldsymbol{\alpha} = [\alpha_1 \ldots \alpha_T]$ their weights satisfying $\alpha_t \geq 0$. The hypotheses $g_t$ are elements of a hypotheses class $G = \{g : \mathbf{x} \mapsto [-1, 1]\}$, which is defined by a base learning algorithm. The ensemble generates the label which is the weighted majority of the votes by

$$\text{sign}(f(\mathbf{x})) \quad \text{where} \quad f(\mathbf{x}) = \sum_t \frac{\alpha_t}{\|\boldsymbol{\alpha}\|_1} g_t(\mathbf{x}). \tag{1}$$

In order to express that $f$ and therefore also the margin $\rho$ depend on $\boldsymbol{\alpha}$ and for ease of notation we define

$$\rho(\mathbf{z}, \boldsymbol{\alpha}) := y f(\mathbf{x}) \text{ where } \mathbf{z} := (\mathbf{x}, y) \text{ and } f \text{ is defined as in (1).} \tag{2}$$

Likewise we use the *normalized* margin:

$$\rho(\boldsymbol{\alpha}) := \min_{1 \leq i \leq m} \rho(\mathbf{z}_i, \boldsymbol{\alpha}) , \tag{3}$$

Ensemble learning methods have to find both, the hypotheses $g_t \in G$ used for the combination and their weights $\boldsymbol{\alpha}$. In the following we will consider only AdaBoost algorithms (including Arcing). For more details see e.g. [4, 2]. The main idea of AdaBoost is to introduce weights $w_t(\mathbf{z}_i)$ on the training patterns. They are used to control the importance of each single pattern in learning a new hypothesis (i.e. while repeatedly running the base algorithm). Training patterns that are difficult to learn (which are misclassified repeatedly) become more important.

The minimization objective of AdaBoost can be expressed in terms of margins as

$$\mathcal{G}(\boldsymbol{\alpha}) := \sum_{i=1}^m \exp(-\|\boldsymbol{\alpha}\|_1 \rho(\mathbf{z}_i, \boldsymbol{\alpha})) . \tag{4}$$

In every iteration, AdaBoost tries to minimize this error by a stepwise maximization of the margin. It is widely believed that AdaBoost tries to maximize the *smallest margin* on the training set [2, 5, 3, 13, 11]. Strictly speaking, however, a general proof is missing. It would imply that AdaBoost asymptotically approximates (up to scaling) the solution of the following linear programming problem over the complete hypothesis set $G$ (cf. [7], assuming a *finite* number of basis hypotheses):

$$\begin{array}{ll} \text{maximize} & \rho \\ \text{subject to} & \rho(\mathbf{z}_i, \boldsymbol{\alpha}) \geq \rho \quad \text{for all } 1 \leq i \leq m \\ & \alpha_t, \rho \geq 0 \quad \text{for all } 1 \leq t \leq |G| \\ & \|\boldsymbol{\alpha}\|_1 = 1 \end{array} \tag{5}$$

Since such a linear program cannot be solved exactly for a infinite hypothesis set in general, it is interesting to analyze approximation algorithms for this kind of problems.

Breiman [2] proposed a modification of AdaBoost – Arc-GV – making it possible to show the asymptotic convergence of $\rho(\boldsymbol{\alpha}^t)$ to the global solution $\rho^{\text{lp}}$:

**Theorem 1 (Breiman [2]).** *Choose $\alpha_t$ in each iteration as*

$$\alpha_t := \operatorname*{argmin}_{\alpha \in [0,1]} \sum_i \exp\left[-\|\boldsymbol{\alpha}^t\|_1 \left(\rho(\mathbf{z}_i, \boldsymbol{\alpha}^t) - \rho(\boldsymbol{\alpha}^{t-1})\right)\right], \tag{6}$$

*and assume that the base learner always finds the hypothesis $g \in G$ which minimizes the weighted training error with respect to the weights. Then*

$$\lim_{t \to \infty} \rho(\boldsymbol{\alpha}^t) = \rho^{\text{lp}}.$$

Note that the algorithm above can be derived from the modified error function

$$\mathcal{G}_{\text{gv}}(\boldsymbol{\alpha}^t) := \sum_i \exp\left[-\|\boldsymbol{\alpha}^t\|_1 \left(\rho(\mathbf{z}_i, \boldsymbol{\alpha}^t) - \rho(\boldsymbol{\alpha}^{t-1})\right)\right]. \tag{7}$$

The question one might ask now is whether to use AdaBoost or rather Arc-GV in practice. Does Arc-GV converge fast enough to benefit from its asymptotic properties? In [12] we conducted experiments to investigate this question. We empirically found that (a) AdaBoost has problems finding the optimal combination if $\rho^{\text{lp}} < 0$, (b) Arc-GV's convergence does not depend on $\rho^{\text{lp}}$, and (c) for $\rho^{\text{lp}} > 0$, AdaBoost usually converges to the maximum margin solution slightly faster than Arc-GV. Observation (a) becomes clear from (4): $\mathcal{G}(\boldsymbol{\alpha})$ will not converge to 0 and $\|\boldsymbol{\alpha}\|_1$ can be bounded by some value. Thus the asymptotic case cannot be reached, whereas for Arc-GV the optimum is always found.

Moreover, the number of iterations necessary to converge to a *good* solution seems to be reasonable, but for a near *optimal* solution the number of iterations is rather high. This implies that for real world hypothesis sets, the number of iterations needed to find an almost optimal solution can become prohibitive, but we conjecture that in practice a reasonably good approximation to the optimum is provided by both AdaBoost and Arc-GV.

## 3 ν-Algorithms

For the LP-AdaBoost [7] approach it has been shown for noisy problems that the generalization performance is usually not as good as the one of AdaBoost [7, 2, 11]. From Theorem 5 in [13] (cf. Theorem 3 on page 5) this fact becomes clear, as the minimum of the right hand side of inequality (cf. (13)) need not necessarily be achieved with a maximum margin. We now propose an algorithm to directly control the number of margin errors and therefore also the contribution of both terms in the inequality separately (cf. Theorem 3). We first consider a small hypothesis class and end up with a linear program – ν-LP-AdaBoost. In subsection 3.2 we then combine this algorithm with the ideas from section 2 and get a new algorithm – ν-Arc – which approximates the ν-LP solution.

### 3.1 ν-LP-AdaBoost

Let us consider the case where we are given a (finite) set $G = \{g : \mathbf{x} \mapsto [-1, 1]\}$ of $T$ hypotheses. To find the coefficients $\boldsymbol{\alpha}$ for the combined hypothesis $f(\mathbf{x})$ we extend the LP-AdaBoost algorithm [7, 11] by incorporating the parameter $\nu$ [15] and solve the following linear optimization problem:

$$
\begin{aligned}
\text{maximize} \quad & \rho - \frac{1}{\nu m} \sum_{i=1}^m \xi_i \\
\text{subject to} \quad & \rho(\mathbf{z}_i, \boldsymbol{\alpha}) \geq \rho - \xi_i \quad \text{for all } 1 \leq i \leq m \\
& \xi_i, \alpha_t, \rho \geq 0 \quad \text{for all } 1 \leq t \leq T \text{ and } 1 \leq i \leq m \\
& \|\boldsymbol{\alpha}\|_1 = 1
\end{aligned}
\tag{8}
$$

This algorithm does not force all margins to be beyond zero and we get a *soft margin* classification (cf. SVMs) with a regularization constant $\frac{1}{\nu m}$. The following proposition shows that $\nu$ has an immediate interpretation:

**Proposition 2 (Rätsch et al. [12]).** *Suppose we run the algorithm given in (8) on some data with the resulting optimal $\rho > 0$. Then*

    1. *$\nu$ upper bounds the fraction of margin errors.*

    2. *$1 - \nu$ upper bounds the fraction of patterns with margin larger than $\rho$.*

Since the slack variables $\xi_i$ only enter the cost function linearly, their absolute size is not important. Loosely speaking, this is due to the fact that for the optimum of the primal objective function, only *derivatives* wrt. the primal variables matter, and the derivative of a linear function is constant.

In the case of SVMs [14], where the hypotheses can be thought of as vectors in some feature space, this statement can be translated into a precise rule for distorting training patterns without changing the solution: we can move them locally orthogonal to a separating hyperplane. This yields a desirable *robustness* property. Note that the algorithm essentially depends on the *number* of outliers, not on the size of the error [15].

### 3.2 The $\nu$-Arc Algorithm

Suppose we have a very large (but finite) base hypothesis class $G$. Then it is difficult to solve (8) as (5) directly. To this end, we propose a new algorithm – $\nu$-Arc – that approximates the solution of (8).

The optimal $\rho$ for fixed margins $\rho(\mathbf{z}_i, \boldsymbol{\alpha})$ in (8) can be written as

$$\rho_\nu(\boldsymbol{\alpha}) := \underset{\rho \in [0,1]}{\operatorname{argmax}} \left( \rho - \frac{1}{\nu m} \sum_{i=1}^{m} (\rho - \rho(\mathbf{z}_i, \boldsymbol{\alpha}))_+ \right). \tag{9}$$

where $(\xi)_+ := \max(\xi, 0)$. Setting $\xi_i := (\rho_\nu(\boldsymbol{\alpha}) - \rho(\mathbf{z}_i, \boldsymbol{\alpha}))_+$ and subtracting $\frac{1}{\nu m} \sum_{i=1}^{m} \xi_i$ from the resulting inequality on both sides yields (for all $1 \leq i \leq m$)

$$\rho(\mathbf{z}_i, \boldsymbol{\alpha}) + \xi_i - \frac{1}{\nu m} \sum_{i=1}^{m} \xi_i \geq \rho_\nu(\boldsymbol{\alpha}) - \frac{1}{\nu m} \sum_{i=1}^{m} \xi_i . \tag{10}$$

Two more substitutions are needed to transform the problem into one which can be solved by the AdaBoost algorithm. In particular we have to get rid of the slack variables $\xi_i$ again by absorbing them into quantities similar to $\rho(\mathbf{z}_i, \boldsymbol{\alpha})$ and $\rho(\boldsymbol{\alpha})$. This works as follows: on the right hand side of (10) we have the objective function (cf. (8)) and on the left hand side a term that depends nonlinearly on $\boldsymbol{\alpha}$. Defining

$$\tilde{\rho}_\nu(\boldsymbol{\alpha}) := \rho_\nu(\boldsymbol{\alpha}) - \frac{1}{\nu m} \sum_{i=1}^{m} \xi_i \quad \text{and} \quad \tilde{\rho}_\nu(\mathbf{z}_i, \boldsymbol{\alpha}) := \rho(\mathbf{z}_i, \boldsymbol{\alpha}) + \xi_i - \frac{1}{\nu m} \sum_{i=1}^{m} \xi_i, \tag{11}$$

which we substitute for $\rho(\boldsymbol{\alpha})$ and $\rho(\mathbf{z}, \boldsymbol{\alpha})$ in (5), respectively, we obtain a new optimization problem. Note that $\tilde{\rho}_\nu(\boldsymbol{\alpha})$ and $\tilde{\rho}_\nu(\mathbf{z}_i, \boldsymbol{\alpha})$ play the role of a *corrected* or *virtual* margin. We obtain a nonlinear min-max problem

$$
\begin{aligned}
\text{maximize} \quad & \tilde{\rho}(\boldsymbol{\alpha}) \\
\text{subject to} \quad & \tilde{\rho}(\mathbf{z}_i, \boldsymbol{\alpha}) \geq \tilde{\rho}(\boldsymbol{\alpha}) \quad \text{for all } 1 \leq i \leq m \\
& \alpha_t \geq 0 \quad \text{for all } 1 \leq t \leq T \\
& \|\boldsymbol{\alpha}\|_1 = 1
\end{aligned}
\tag{12}
$$

which Arc-GV can solve approximately (cf. section 2). Hence, by replacing the margin $\rho(\mathbf{z}, \boldsymbol{\alpha})$ by $\tilde{\rho}(\mathbf{z}, \boldsymbol{\alpha})$ in equation (4) and the other formulas for Arc-GV (cf. [2]),

we obtain a new algorithm which we refer to as *ν-Arc*.

We can now state interesting properties for $\nu$-Arc by using Theorem 5 of [13] that bounds the generalization error $R(f)$ for ensemble methods. In our case $R_\rho(f) \leq \nu$ by construction (i.e. the number of patterns with a margin smaller than $\rho$, cf. Proposition 2), thus we get the following simple reformulation of this bound:

**Theorem 3.** *Let $p(\mathbf{x}, y)$ be a distribution over $\mathcal{X} \times [-1, 1]$, and let $X$ be a sample of $m$ examples chosen iid according to $p$. Suppose the base-hypothesis space $G$ has VC dimension $h$, and let $\delta > 0$. Then with probability at least $1 - \delta$ over the random choice of the training set $X, Y$, every function $f$ generated by $\nu$-Arc, where $\nu \in (0, 1)$ and $\rho_\nu > 0$, satisfies the following bound:*

$$R(f) \leq \nu + \sqrt{\frac{c}{m} \left( \frac{h \log^2 (m/h)}{\rho_\nu^2} + \log \left( \frac{1}{\delta} \right) \right)}. \tag{13}$$

So, for minimizing the right hand side we can tradeoff between the first and the second term by controlling an easily interpretable regularization parameter $\nu$.

## 4 Experiments

We show a set of toy experiments to illustrate the general behavior of $\nu$-Arc. As base hypothesis class $G$ we use the RBF networks of [11], and as data a two-class problem generated from several 2D Gauss blobs (cf. **Banana shape** dataset from http://www.first.gmd.de/~data/banana.html.). We obtain the following results:

- $\nu$-Arc leads to approximately $\nu m$ patterns that are effectively used in the training of the base learner: Figure 1 (left) shows the fraction of patterns that have high average weights during the learning process (i.e. $\sum_{t=1}^T w_t(\mathbf{z}_i) > 1/2m$). We find that the number of the latter increases (almost) linearly with $\nu$. This follows from (11) as the (soft) margin of patterns with $\rho(\mathbf{z}, \boldsymbol{\alpha}) < \rho_\nu$ is set to $\rho_\nu$ and the weight of those patterns will be the same.

- The (estimated) test error, averaged over 10 training sets, exhibits a rather flat minimum in $\nu$ (Figure 1 (lower)). This indicates that just as for $\nu$-SVMs, where corresponding results have been obtained, $\nu$ is a well-behaved parameter in the sense that a slight misadjustment it is not harmful.

- $\nu$-Arc leads to the fraction $\nu$ of margin errors (cf. dashed line in Figure 1) exactly as predicted in Proposition 2.

- Finally, a good value of $\nu$ can already be inferred from prior knowledge of the expected error. Setting it to a value similar to the latter provides a good starting point for further optimization (cf. Theorem 3).

Note that for $\nu = 1$, we recover the Bagging algorithm (if we used bootstrap samples), as the weights of all patterns will be the same ($w_t(\mathbf{z}_i) = 1/m$ for all $i = 1, \ldots, m$) and also the hypothesis weights will be constant ($\alpha_t \sim 1/T$ for all $t = 1, \ldots, T$).

Finally, we present a small comparison on ten benchmark data sets obtained from the UCI [1] benchmark repository (cf. http://ida.first.gmd.de/~raetsch/data/benchmarks.html). We analyze the performance of single RBF networks, AdaBoost, $\nu$-Arc and RBF-SVMs. For AdaBoost and $\nu$-Arc we use RBF networks [11] as base hypothesis. The model parameters of RBF (number of centers etc.), $\nu$-Arc ($\nu$) and SVMs ($\sigma, C$) are optimized using 5-fold cross-validation. More details on the experimental setup can

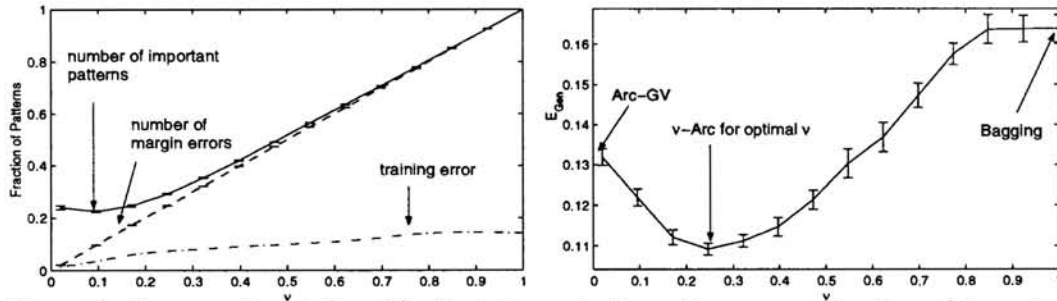

Figure 1: Toy experiment ($\sigma = 0$): the left graph shows the average fraction of *important* patterns, the av. fraction of margin errors and the av. training error for different values of the regularization constant $\nu$ for $\nu$-Arc. The right graph shows the corresponding generalization error. In both cases, the parameter $\nu$ allows us to reduce the test errors to values much lower than for the hard margin algorithm (for $\nu = 0$ we recover Arc-GV/AdaBoost, and for $\nu = 1$ we get Bagging.)

be found in [11]. Fig. 1 shows the generalization error estimates (after averaging over 100 realizations of the data sets) and the confidence interval. The results of the best classifier and the classifiers that are not significantly worse are set in bold face. To test the significance, we used a $t$-test ($p = 80\%$). On eight out of the ten data sets, $\nu$-Arc performs significantly better than AdaBoost. This clearly shows the superior performance of $\nu$-Arc for noisy data sets and supports this soft margin approach for AdaBoost. Furthermore, we find comparable performances for $\nu$-Arc and SVMs. In three cases the SVM performs better and in two cases $\nu$-Arc performs best. Summarizing, AdaBoost is useful for low noise cases, where the classes are separable. $\nu$-Arc extends the applicability of boosting to problems that are difficult to separate and should be applied if the data are noisy.

## 5 Conclusion

We analyzed the AdaBoost algorithm and found that Arc-GV and AdaBoost are efficient for approximating the solution of non-linear min-max problems over huge hypothesis classes. We re-parameterized the $\text{LP}_{Reg}$-AdaBoost algorithm (cf. [7, 11]) and introduced a new regularization constant $\nu$ that controls the fraction of patterns inside the margin area. The new parameter is highly intuitive and has to be optimized only on a fixed interval $[0, 1]$.
Using the fact that Arc-GV can approximately solve min-max problems, we found a formulation of Arc-GV – $\nu$-Arc – that implements the $\nu$-idea for Boosting by defining an appropriate *soft margin*. The present paper extends previous work on regularizing boosting (DOOM [9], AdaBoost$_{Reg}$ [11]) and shows the utility and flexibility of the soft margin approach for AdaBoost.

| | RBF | AB | $\nu$-Arc | SVM |
|---|---|---|---|---|
| Banana | 10.8 ± 0.06 | 12.3 ± 0.07 | **10.6 ± 0.05** | 11.5 ± 0.07 |
| B.Cancer | 27.6 ± 0.47 | 30.4 ± 0.47 | **25.8 ± 0.46** | **26.0 ± 0.47** |
| Diabetes | 24.3 ± 0.19 | 26.5 ± 0.23 | **23.7 ± 0.20** | **23.5 ± 0.17** |
| German | 24.7 ± 0.24 | 27.5 ± 0.25 | 24.4 ± 0.22 | **23.6 ± 0.21** |
| Heart | 17.6 ± 0.33 | 20.3 ± 0.34 | **16.5 ± 0.36** | **16.0 ± 0.33** |
| Ringnorm | **1.7 ± 0.02** | 1.9 ± 0.03 | **1.7 ± 0.02** | **1.7 ± 0.01** |
| F.Sonar | 34.4 ± 0.20 | 35.7 ± 0.18 | 34.4 ± 0.19 | **32.4 ± 0.18** |
| Thyroid | **4.5 ± 0.21** | **4.4 ± 0.22** | **4.4 ± 0.22** | 4.8 ± 0.22 |
| Titanic | 23.3 ± 0.13 | **22.6 ± 0.12** | 23.0 ± 0.14 | **22.4 ± 0.10** |
| Waveform | 10.7 ± 0.11 | 10.8 ± 0.06 | **10.0 ± 0.07** | **9.9 ± 0.04** |

Table 1: Generalization error estimates and confidence intervals. The best classifiers for a particular data set are marked in bold face (see text).

We found empirically that the generalization performance in $v$-Arc depends only slightly on the choice of the regularization constant. This makes model selection (e.g. via cross-validation) easier and faster.

Future work will study the detailed regularization properties of the regularized versions of AdaBoost, in particular in comparison to $v$-LP Support Vector Machines.

**Acknowledgments:** Partial funding from DFG grant (Ja 379/52) is gratefully acknowledged. This work was done while AS and BS were at GMD FIRST.

# References

[1] C. Blake, E. Keogh, and C. J. Merz. UCI repository of machine learning databases, 1998. http://www.ics.uci.edu/~mlearn/MLRepository.html.

[2] L. Breiman. Prediction games and arcing algorithms. Technical Report 504, Statistics Department, University of California, December 1997.

[3] M. Frean and T. Downs. A simple cost function for boosting. Technical report, Dept. of Computer Science and Electrical Eng., University of Queensland, 1998.

[4] Y. Freund and R. E. Schapire. A decision-theoretic generalization of on-line learning and an application to boosting. In *Computational Learning Theory: Eurocolt '95*, pages 23–37. Springer-Verlag, 1995.

[5] Y. Freund and R. E. Schapire. A decision-theoretic generalization of on-line learning and an application to boosting. *J. of Comp. & Syst. Sc.*, 55(1):119–139, 1997.

[6] J. Friedman, T. Hastie, and R. Tibshirani. Additive logistic regression: a statistical view of boosting. Technical report, Stanford University, 1998.

[7] A. Grove and D. Schuurmans. Boosting in the limit: Maximizing the margin of learned ensembles. In *Proc. of the 15th Nat. Conf. on AI*, pages 692–699, 1998.

[8] Y. LeCun, L. D. Jackel, L. Bottou, C. Cortes, J. S. Denker, H. Drucker, I. Guyon, U. A. Müller, E. Säckinger, P. Simard, and V. Vapnik. Learning algorithms for classification: A comparison on handwritten digit recognition. *Neural Networks*, pages 261–276, 1995.

[9] L. Mason, P. L. Bartlett, and J. Baxter. Improved generalization through explicit optimization of margins. *Machine Learning*, 1999. to appear.

[10] J. R. Quinlan. Boosting first-order learning (invited lecture). *Lecture Notes in Computer Science*, 1160:143, 1996.

[11] G. Rätsch, T. Onoda, and K.-R. Müller. Soft margins for AdaBoost. Technical Report NC-TR-1998-021, Department of Computer Science, Royal Holloway, University of London, Egham, UK, 1998. To appear in Machine Learning.

[12] G. Rätsch, B. Schökopf, A. Smola, S. Mika, T. Onoda, and K.-R. Müller. Robust ensemble learning. In A.J. Smola, P.L. Bartlett, B. Schölkopf, and D. Schuurmans, editors, *Advances in LMC*, pages 207–219. MIT Press, Cambridge, MA, 1999.

[13] R. Schapire, Y. Freund, P. L. Bartlett, and W. Sun Lee. Boosting the margin: A new explanation for the effectiveness of voting methods. *Annals of Statistics*, 1998. (Earlier appeared in: D. H. Fisher, Jr. (ed.), Proc. ICML97, M. Kaufmann).

[14] B. Schölkopf, C. J. C. Burges, and A. J. Smola. *Advances in Kernel Methods — Support Vector Learning*. MIT Press, Cambridge, MA, 1999.

[15] B. Schölkopf, A. Smola, R. C. Williamson, and P. L. Bartlett. New support vector algorithms. *Neural Computation*, 12:1083 – 1121, 2000.

[16] H. Schwenk and Y. Bengio. Training methods for adaptive boosting of neural networks. In Michael I. Jordan, Michael J. Kearns, and Sara A. Solla, editors, *Advances in Neural Inf. Processing Systems*, volume 10. The MIT Press, 1998.

[17] V. Vapnik. *The Nature of Statistical Learning Theory*. Springer Verlag, New York, 1995.
